# Probabilistic Inference and Differential Privacy

**Oliver Williams**
Microsoft Research
Mountain View, CA 94043
olliew@microsoft.com

**Frank McSherry**
Microsoft Research
Mountain View, CA 94043
mcsherry@microsoft.com

## Abstract

We identify and investigate a strong connection between probabilistic inference and differential privacy, the latter being a recent privacy definition that permits only indirect observation of data through noisy measurement. Previous research on differential privacy has focused on designing measurement processes whose output is likely to be useful on its own. We consider the potential of applying probabilistic inference to the measurements and measurement process to derive posterior distributions over the data sets and model parameters thereof. We find that probabilistic inference can improve accuracy, integrate multiple observations, measure uncertainty, and even provide posterior distributions over quantities that were not directly measured.

## 1 Introduction

There has recently been significant interest in the analysis of data sets whose individual records are too sensitive to expose directly, examples of which include medical information, financial data, and personal data from social networking sites. Data like these are rich sources of information from which models could be learned for a variety of important applications. Although agencies with the resources to collate such data are unable to grant outside parties direct access to them, they may be able to safely release aggregate statistics of the data set. Progress in this area has so far been driven by researchers inventing sophisticated learning algorithms which are applied directly to the data and output model parameters which can be proven to respect the privacy of the data set. Proving these privacy properties requires an intricate analysis of each algorithm on a case-by-case basis. While this does result in many valuable algorithms and results, it is not a scalable solution for two reasons: first, to solve a new learning problem, one must invent and analyze a new privacy-preserving algorithm; second, one must then convince the owner of the data to run this algorithm. Both of these steps are challenging.

In this paper, we show a natural connection between differential privacy, one of the leading privacy definitions, and probabilistic inference. Specifically, differential privacy exposes the conditional distribution of its observable outputs given any input data set. Combining the conditional distributions of differentially-private observations with generative models for the data permits new inferences about the data without the need to invent and analyze new differentially-private computations. In some cases, one can rely on previously reported differentially private measurements. When this is not sufficient, one can use off-the-shelf differentially-private "primitives" pre-sanctioned by owners of the data. As well as this flexibility, probabilistic inference can improve the accuracy of existing approaches, provide a measure of uncertainty in any predictions made, combine multiple observations in a principled way, and integrate prior knowledge about the data or parameters.

The following section briefly introduces differential privacy. In Section 3 we explore the marginal likelihood of the differentially-private observations given generative model parameters for the data. In general this likelihood consists of a high-dimensional integration over the space of all data sets, however we show that for a rich subclass of differentially private computations this distribution can

be efficiently approximated via upper and lower bounds, derived using variational techniques. Section 4 shows several experimental results validating our hypothesis that probabilistic inference can be fruitfully applied to differentially-private computation. In particular, we show how the application of principled, probabilistic inference to measurements made by an existing, heuristic algorithm for logistic regression improves performance, as well as providing confidence on the predictions made.

## 1.1 Related work

There is a substantial amount of research on privacy, and differential privacy in particular, connected with machine learning and statistics. Nonetheless, we are unaware of any research that uses exact knowledge of the conditional distribution over outputs given inputs to perform inference over model parameters, or other features of the data. Much of the existing statistical literature is concerned with identifying cases when the differentially-private observations are "as good" as traditional statistical estimators, in terms of efficiency [1], power [2], and minimax rates [3], and also robust estimators [4]. Instead, we are concerned with the cases where it is valuable to acknowledge and manage the uncertainty in the observations. As we demonstrate experimentally, such cases abound.

Chaudhuri and Monteleoni [5, 6] introduced the NIPS community to the problem of differentially-private logistic regression. Although we will also consider the problem of logistic regression (and compare our findings with theirs) we should stress that the aim of the paper is not specifically to attack the problem of logistic regression. Rather, the problem serves as a good example where prior work on differentially-private logistic regression can be improved through probabilistic inference.

## 2 Differential Privacy

Differential privacy [7] applies to randomized computations executed against a dataset and returning an aggregate result for the entire set. It prevents inference about specific records by requiring that the result of the computation yield nearly identical distributions for similar data sets. Formally, a randomized computation $M$ satisfies $\epsilon$-differential privacy if for any two possible input data sets $A$ and $B$, and any subset of possible outputs $S$,

$$P(M(A) \in S) \quad \leq \quad P(M(B) \in S) \times \exp(\epsilon \times |A \ominus B|) \,, \tag{1}$$

where $A \ominus B$ is the set of records in $A$ or $B$, but not both. When $A \ominus B$ is small, the relative bound on probabilities limits the inference an attacker can make about whether the true underlying data were actually $A$ or $B$. Inferences about the presence, absence, or specific values of individual records are strongly constrained.

One example of a differentially private computation is the *exponential mechanism*[8], characterized by a function $\phi : D^n \times R \to \mathbb{R}$ scoring each pair of data set and possible result with a real value. When the $\phi$ function satisfies $|\ln \phi(z, A) - \ln \phi(z, B)| \leq |A \ominus B|$ for all $z$, the following distribution satisfies $2\epsilon$-differential privacy:

$$P(M(X) = z) \quad = \quad \frac{\phi(z, X)^\epsilon}{\sum_{z' \in Z} \phi(z', X)^\epsilon} \tag{2}$$

The exponential mechanism is fully general for differential privacy; any differentially-private mechanism $M$ can be encoded in a $\phi$ function using the density of $M(X)$ at $z$.

While any differentially-private mechanism can be expressed as a $\phi$ function, verifying that a function $\phi$ satisfies the constraint $|\ln \phi(z, A) - \ln \phi(z, B)| \leq |A \ominus B|$ is generally not easy, and requires some form of proof on a case by case basis. One that does not require a specialized proof, is when the $\phi$ functions can be expressed as $\phi(z, X) = \prod_i \phi(z, x_i)$. This subclass is useful practically, as data providers can ensure differential privacy by clamping each $\phi(z, x)$ value to the range $[e^{-1}, e^{+1}]$, without having to understand the $\phi$ function. We will refer to this subclass as the *factored exponential mechanism*.

As we can see from the definition of the exponential mechanism, a differentially-private mechanism draws its guarantees from its inherent randomness, rather than from secrecy about its specification. Although differential privacy has many other redeeming features, it is this feature alone that we

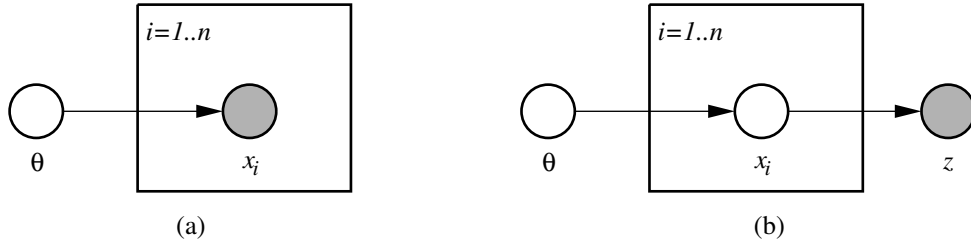

Figure 1: **Graphical models.** *(a) If the data $X = \{x_i\}$ are directly observable (shaded nodes), the canonical learning task is to infer the posterior over $\theta$ given a model relating $X$ and $\theta$. (b) In the private setting, the data are not observable; instead we observe the private measurement $z$, related to $X$ by a known measurement process.*

exploit in the remainder of the work. By the same token, although there are many other privacy definitions with varying guarantees, we can apply inference to any definition exhibiting one key feature: an explicit probabilistic relationship between the input data sets and output observations.

## 3 Inference and privacy

Differential privacy limits what can be inferred about a single record in a data set, but does not directly limit inference about larger scale, aggregate properties of data sets. For example, many tasks in machine learning and statistics infer global parameters describing a model of the data set without explicit dependence on any single record, and we may still expect to be see a meaningful relationship between differentially-private measurements and model parameters.

One way to model a data set is to propose a generative probabilistic model for the data $p(X|\theta)$. In Figure 1(a) we show a graphical model for the common case, in which we seek to infer the parameters $\theta$ given the observed iid data $X = \{x_i\}$. In Figure 1(b) we see a graphical model for the case considered in this paper, in which the data are not directly observed due to privacy. Instead, information about $X$ is revealed by the measurement $z$, which is generated from $X$ according to a known conditional distribution $p(z|X)$, for example as given in (2). We therefore reason about $\theta$ via the *marginal likelihood*

$$p(z|\theta) = \int dX \ p(X|\theta)p(z|X). \tag{3}$$

Armed with the marginal likelihood, it is possible to bring all the techniques of probabilistic inference to bear. This will generally include adding a prior distribution over $\theta$, and combining multiple measurements to form a posterior

$$p(\theta|z_1 \ldots z_m, \pi) = p(\theta|\pi) \prod_j p(z_j|\theta) \tag{4}$$

where $\pi$ stands for any non-private information about $\theta$ we may have available.

While this is superficially clean, there is a problem: the integration in (3) is over the space of all data sets and is therefore challenging to compute whenever it cannot be solved analytically. Section 4 will show some results in which we tackle this head-on via MCMC, however this only works for data sets of moderate size. Therefore, the remainder of this section is devoted to the development of several bounds on the marginal likelihood for cases in which the measurement is generated via the factored exponential mechanism. These bounds can be computed without requiring an integration over all $X$.

### 3.1 Factored exponential mechanism

The factored exponential mechanism of Section 2 is a special case of differentially-private mechanism that admits efficient approximation of the marginal likelihood. We will be able to use the independence in $p(X|\theta) = \prod_i p(x_i|\theta)$ and $\phi(z, X) = \prod_i \phi(z, x_i)$ to factorize lower and upper

bounds on the integral (3), resulting in a small number of integrals over only the domain of records, rather than the domain of data sets. As we will see, the bounds are often quite tight.

$$p(z|\theta) \ge \left( \sum_{z' \in Z} \left( \int dx\, p(x|\theta)\, \frac{\phi(z', x)^\epsilon}{\phi(z, x)^\epsilon} \right)^n \right)^{-1} \tag{5a}$$

$$p(z|\theta) \le e^{-H[q]} \left( \int dx\, p(x|\theta) \frac{\phi(z, x)^\epsilon}{\prod_{z' \in Z} \phi(z', x)^{\epsilon q(z')}} \right)^n \tag{5b}$$

where the upper bound is defined in terms of a variational distribution $q(z)$ [9] such that $\sum_{z \in} q(z) = 1$. $H[q]$ is the Shannon entropy of $q$. Notice that the integrations appearing in either bound are over the space of a single record in a data set and not over the entire dataset as they were in (3).

**Proof of lower bound**

To prove the lower bound, we will apply Jensen's inequality with the function $f(x) = 1/x$ to the marginal likelihood of the exponential mechanism.

$$\int dX\, p(X|\theta) \left( \frac{\phi(z, X)^\epsilon}{\sum_{z' \in Z} \phi(z', X)^\epsilon} \right) \ge \left( \int dX\, p(X|\theta) \sum_{z' \in Z} \frac{\phi(z', X)^\epsilon}{\phi(z, X)^\epsilon} \right)^{-1}$$

$$= \left( \sum_{z' \in Z} \int dX\, p(X|\theta) \frac{\phi(z', X)^\epsilon}{\phi(z, X)^\epsilon} \right)^{-1}$$

which now factorizes, as

$$\int dx_1 \int dx_2 \ldots \int dx_n \prod_i p(x_i|\theta) \frac{\phi(z', x_i)^\epsilon}{\phi(z, x_i)^\epsilon} = \prod_i \left( \int dx_i\, p(x_i|\theta) \frac{\phi(z', x_i)^\epsilon}{\phi(z, x_i)^\epsilon} \right)$$

$$= \left( \int dx\, p(x|\theta) \frac{\phi(z', x)^\epsilon}{\phi(z, x)^\epsilon} \right)^n.$$

■

**Proof of upper bound**

We can lower bound the normalizing term $\sum_{z' \in Z} \phi(z', X)^\epsilon$ in (2) by introducing a variational distribution $q(z)$, and applying Jensen's inequality with the function $f(x) = \log x$.

$$\sum_{z' \in Z} \phi(z', X)^\epsilon = \exp \log \sum_{z' \in Z} \frac{q(z')}{q(z')} \phi(z', X)^\epsilon$$

$$\ge \exp(H[q]) + \exp \left( \sum_{z' \in Z} q(z') \log \phi(z', X)^\epsilon \right).$$

Applying this bound to the marginal likelihood gives us the bound

$$\int dX\, p(X|\theta) \frac{\phi(z, X)^\epsilon}{\sum_{z' \in Z} \phi(z', X)^\epsilon} \le e^{-H[q]} \int dX\, p(X|\theta) \frac{\phi(z, X)^\epsilon}{\prod_{z' \in Z} \phi(z', X)^{\epsilon q(z')}}$$

$$= e^{-H[q]} \int dX \prod_i \left( p(x_i|\theta) \frac{\phi(z, x_i)^\epsilon}{\prod_{z' \in Z} \phi(z', x_i)^{\epsilon q(z')}} \right)$$

$$= e^{-H[q]} \left( \int dx\, p(x|\theta) \frac{\phi(z, x)^\epsilon}{\prod_{z' \in Z} \phi(z', x)^{\epsilon q(z')}} \right)^n.$$

■

While the upper bound is true for any $q$ distribution, the tightest bound is found for the $q$ which minimizes the bound.

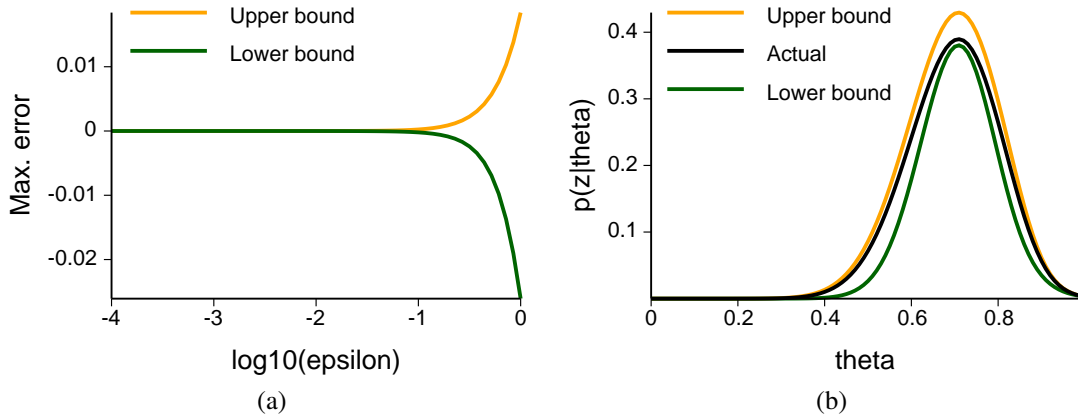

Figure 2: **Error in upper and lower bounds for coin-flipping problem.** *(a) For each epsilon, we plot the maximum across all $\theta$ of the error between the true distribution and each of the upper and lower bounds is plotted. (b) For $n = 100$ and $\epsilon = 0.5$, we show the shape of the upper bound, lower bound, and true distribution when differentially-private measurement returned was $z = 0.7$.*

### 3.1.1   Chosing a $\phi$ function

The upper and lower bounds in (5) are true for any admissible $\phi$ function, but leave unanswered the question of what to chose in this rôle. In the absence of privacy we might try to find a good fit for the parameters $\theta$ by maximum likelihood. In the private setting this is not possible because the data are not directly observable, but the output of the factored exponential mechanism has a very similar form:

$$\text{Max likelihood:} \qquad \theta^* = \arg\max_{\theta \in \Theta} \prod_i p(x_i|\theta) \tag{6a}$$

$$\text{Exp. mechanism:} \qquad z^* = \text{noisy}\max_{z \in Z} \prod_i \phi(z, x_i)^\epsilon \tag{6b}$$

where noisy $\max_{z \in Z} f(z)$ samples from $\frac{f(z)}{\sum_{z' \in Z} f(z')}$. By making the analogy between (6a) and (6b), we might let $z$ range over elements of $\Theta$ (or a finite subset), and take $\phi(z, x_i)$ to be the likelihood of $x_i$ under parameters $z$. The exponential mechanism is then likely to choose parameters $z$ that fit the data well, informing us that the posterior over $\theta$ is likely in the vicinity of $z$. For $\phi$ to be admissible, we must clamp very small values of $\phi$ up to $1/e$, limiting the ability of very poorly fit records to influence our decisions strongly.

### 3.2   Evaluation of the bounds

To demonstrate the effectiveness of these bounds we consider a problem in which it is possible to analytically compute the marginal likelihood. This is the case in which the database $X$ contains a set of Boolean values corresponding to independent samples from a Bernoulli distribution with probability $\theta$

$$p(x|\theta) = \theta^x (1-\theta)^{(1-x)}. \tag{7}$$

For our test, we took $Z$ to be the nine multiples of 0.1 between 0.1 and 0.9, and $\log \phi(z, x_i) = [\log p(x|\theta)]_{\log 0.1}^{\log 0.9}$ that is, the log likelihood clammped such that $\phi(z, x)$ lies in the interval $[e^{-1}, e^{+1}]$, as required by privacy.

We see in figure 2a that the error in both the upper and lower bounds, across the entire density function, is essentially zero for small epsilon. As epsilon increases the bounds deteriorate, but we are most interested in the case of small values of epsilon, where privacy guarantees are meaningfully strong. Figure 2b shows the shape of the two bounds, and the true density between, for epsilon = 0.5. This large value was chosen as it is in the region for which the bounds are less tight and the difference between the bounds and the truth can be seen.

The upper bound is defined in terms of a variational distribution $q$. For these experiments $q$ was approximately minimized by setting $q(z) \propto \exp\left(n \int dx \, p(x|\theta) \log \phi(z, x)\right)$. In general, however, these (and other) test show that both bounds are equally good for reasonable values of $\epsilon$ and we therefore use the lower bound for the experiments in this paper, since it is simpler to compute.

## 4 Experiments

We consider two scenarios for the experimental validation of the utility of probabilistic inference. First, we consider applying probabilistic inference to an existing differentially-private computation, specifically a logistic regression heuristic taken from a suite of differentially-private algorithms. The heuristic is not representable in the factored exponential mechanism, and as such we must attempt to approximate the full integral over the space of data sets directly. In our second experiment, we choose a problem and measurement process appropriate for the factored exponential mechanism, principal components analysis, previously only ever addressed through noisy observation of the covariance matrix.

### 4.1 Logistic Regression

To examine the potential of probabilistic inference to improve the quality of existing differentially-private computations, we consider a heuristic algorithm for logistic regression included in the Privacy Integrated Queries distribution [10]. This heuristic uses a noisy sum primitive to repeatedly compute and step in the direction of an approximate gradient. When the number of records is large compared to the noise introduced, the approximate gradient is relatively accurate, and the algorithm performs well. When the records are fewer or the privacy requirements demand more noise, its performance suffers. Probabilistic inference has the potential to improve performance by properly integrating the information extracted from the data across the multiple gradient measurements and managing the uncertainty associated with the noisy measurements.

We test our proposals against three synthetic data sets (CM1 and CM2 from [5] and one of our own: SYNTH) and two data sets from the UCI repository (PIMA and ADULT) [11]. Details of these data sets appear in table 4.1. The full ADULT data set was split into training and test sets, chosen so as to force the marginal frequency of positive and negative examples to 50%.

|  | SYNTH | CM1 | CM2 | PIMA | ADULT |
|---:|:---:|:---:|:---:|:---:|:---:|
| *Records* | 1000 | 17500 | 17500 | 691 | 16000 |
| *Dimensions* | 4 | 10 | 10 | 8 | 6 |
| *Positive examples* | 497 | 8770 | 8694 | 237 | 7841 |
| *Test set records* | 1000 | 17500 | 17500 | 767* | 8000 |

Table 1: **Data sets used and their statistics.** *Attribute values in SYNTH are sampled uniformly from a hypercube of unit volume, centered at the origin. CM1 and CM2 are both sampled uniformly at random for the surface of the unit hypersphere in 10 dimensions; CM1 is linearly separable, whereas CM2 is not (see [5]). PIMA and ADULT are standard data sets [11] containing diabetes records, and census data respectively, both of which correspond to the types of data one might expect to be protected by differential privacy. The total PIMA data set is so small that we reused the full data set as test data (indicated by *).*

#### 4.1.1 Error Rates and Log-Likelihood

Tables 2 and 3 report the classification accuracy of several approaches when the privacy parameter $\epsilon$ is set to 0.1 and 1.0 respectively. These results are computed from 50 executions of the heuristic gradient descent algorithm.

We can see a trend of general improvement from the heuristic approach to the probabilistic inference, both in terms of the average error rate and the standard deviation. For the CM1 and CM2 data sets at epsilon = 0.1, we see substantial improvement over the reported results of [5]. Please note that

|  | SYNTH | CM1 | CM2 | PIMA | ADULT |
|---|---|---|---|---|---|
| *Heuristic* | $37.40 \pm 15.75$ | $3.93 \pm 1.57$ | $9.32 \pm 1.18$ | $44.26 \pm 8.50$ | $43.15 \pm 7.85$ |
| *Inference* | $29.14 \pm 5.54$ | $2.72 \pm 0.84$ | $8.84 \pm 0.79$ | $45.70 \pm 6.31$ | $36.07 \pm 6.32$ |
| *Benchmark* | 16.40 | 0.00 | 5.40 | 19.48 | 26.09 |
| *NIPS 08 [5]* |  | $14.26 \pm 12.84$ | $19.03 \pm 11.05$ |  |  |

Table 2: **Error Rates with $\epsilon$ = 0.1** *All measurements are in per cent; errors are reported as the mean $\pm$ one standard deviation computed from 50 independent executions with random starting points.* **Heuristic** *corresponds to the last estimate made by noisy gradient ascent.* **Inference** *entries correspond to the expected error, computed over the approximate posterior for $\theta$ found via MCMC.* **Benchmark** *is the best maximum likelihood solution found by gradient ascent* when the data are directly observable *and forms a baseline for expected performance.* **NIPS08** *corresponds the the results given in [5]; these values were copied from that paper and are provided for comparison.*

|  | SYNTH | CM1 | CM2 | PIMA | ADULT |
|---|---|---|---|---|---|
| *Heuristic* | $17.31 \pm 1.12$ | $0.00 \pm 0.00$ | $5.67 \pm 0.19$ | $35.67 \pm 6.45$ | $31.30 \pm 4.16$ |
| *Inference* | $17.16 \pm 0.94$ | $0.01 \pm 0.02$ | $5.69 \pm 0.13$ | $36.47 \pm 8.56$ | $29.36 \pm 1.31$ |
| *Benchmark* | 16.40 | 0.00 | 5.40 | 19.48 | 26.09 |

Table 3: **Error Rates with $\epsilon$ = 1.0** *All measurements are in per cent; see caption for table 2.*

the experiments were run on different data than in [5] drawn from the same distribution, and that different numbers of repetitions were used in [5] for the computation of the standard deviation and mean.

### 4.1.2 Exchanging Iterations for Accuracy

The heuristic gradient ascent algorithm has an important configuration parameter determining the number of iterations of ascent, and consequently the accuracy permitted in each round (which must be lower if more rounds are to be run, to keep the cumulative privacy cost constant). The performance of the algorithm can be very sensitive to this parameter, as too few iterations indicate too little about the data, and too many render each iteration meaningless. In Figure 3a we consider several parameterizations of the heuristic, taking varying numbers of steps with varying degrees of accuracy in each step. Each colored path describes an execution with a fixed level of accuracy in each iteration, and all are plotted on the common scale of total privacy consumption. All of these paths roughly describe a common curve, suggesting that careful configuration is not required for these approaches: probabilistic inference appears to extract an amount of information that depends mainly on the total privacy consumption, and less on the specific details of its collection. This experiment was performed on the CM2 data set and the corresponding result from [5] is indicated by the 'X'.

### 4.1.3 Integrating Auxiliary Information

To further demonstrate the power of the probabilistic inference approach, we consider the plausible scenario in which we are provided with a limited number of additional data points, obtained *without* privacy protection (for example, if we independently run a small survey of our own). These additional samples are easily incorporated into the graphical model by adding them as descendants of $\theta$ in figure 1b. Figure 3b shows how the performance on SYNTH (which contains 1000 data points) improves, as the quantity of additional examples increases. Even with very few additional examples, probabilistic inference is capable of exploiting this information and performance improves dramatically.

### 4.2 Principal components

To demonstrate inference on another model, and to highlight the applicability of the factored exponential mechanism, we consider the problem of probabilistically finding the first principal compo-

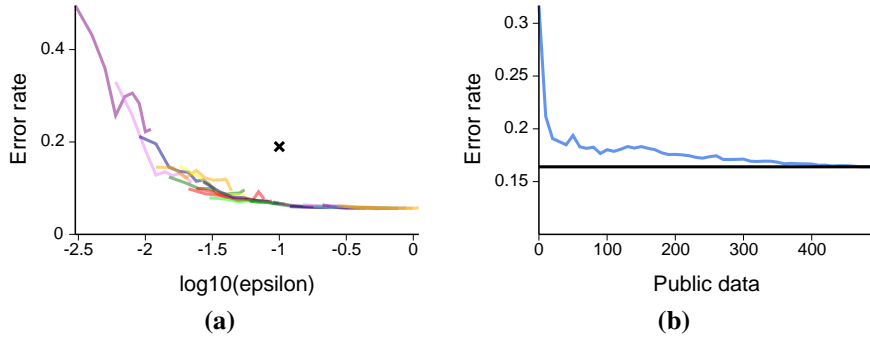

**(a)**                                    **(b)**

Figure 3: **(a) Paths of varying $\epsilon$. (b) Incorporating non-private observations** *A compelling benefit of probabilistic inference is how easily alternate sources of information are added. The horizontal line indicates the performance of the benchmark maximum likelihood solution computed from the data without privacy.*

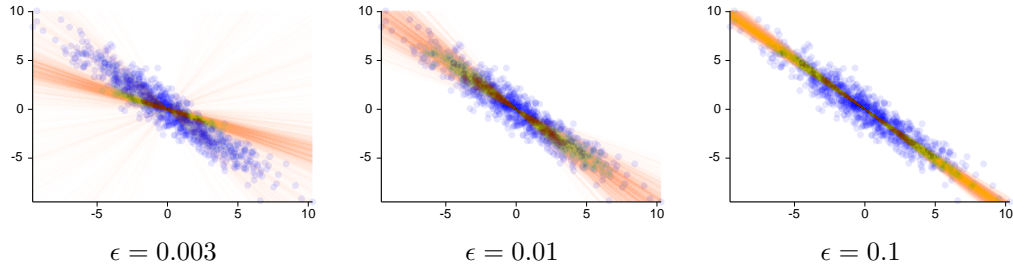

$\epsilon = 0.003$                  $\epsilon = 0.01$                  $\epsilon = 0.1$

Figure 4: **Posterior distribution as a function of** $\epsilon$ *The same synthetic data set under differentially-private measurements with varying epsilon. For each measurement, 1000 samples of the full posterior over $\theta$ are drawn and overlaid on this figure to indicate the modes and concentration of the density. The posterior is noticeably more concentrated and accurate as epsilon increases.*

nent of a data set where we model the data as iid draws from a Gaussian

$$p(x|\theta) = \mathcal{N}(0, \theta\theta^{\mathrm{T}} + \sigma^2 I). \tag{8}$$

An important advantage of our approach is its ability to capture uncertainty in the parameters and act accordingly. Figure 4 demonstrates three instances of inference applied to the same data set with three different values of $\epsilon$. As $\epsilon$ increases, the concentration of the posterior over the parameters increases. We stress that the posterior and its concentration are returned to the analyst; each image is the result of a single differentially-private measurement, rather than a visualization of multiple runs. The measurement associated with $\epsilon = 0.003$ is revealing as it corresponds to the off-axis mode of the posterior. Although centered on this incorrect answer, the posterior indicates lack of confidence, and there is non-negligible mass over the correct answer.

## 5   Conclusions

Most work in the area of learning from private data forms an *intrinsic* analysis. That is, a complex algorithm is run by the owner of the data, directly on that data, and a single output is produced which appropriately indicates the desired parameters (modulo noise). In contrast, this paper has shown that it is possible to do a great deal with an *extrinsic* analysis, where standard, primitive, measurements are made against the data, and a posterior over model parameters is inferred post hoc.

This paper brings together two complementary lines of research: the design and analysis of differentially-private algorithms, and probabilistic inference. Our primary goal is not to weigh-in on new differentially-private algorithms, nor to find new methods for probabilistic inferences – it is to present the observation that the two approaches are complementary in way that can be mutually enriching.

## References

[1] A. Smith. Efficient, differentially private point estimators. 2008. arXiv:0809.4794.

[2] A. Slavkovic and D. Vu. Differential privacy for clinical trial data: Preliminary evaluations. In *Proceedings of the International workshop on Privacy Aspects of Data Mining, PADM09*, 2009.

[3] L. Wasserman and S. Zhou. A statistical framework for differential privacy. *Journal of the American Statistical Association*, 105(489):375–389, 2010.

[4] C. Dwork and J. Lei. Differential privacy and robust statistics. In *STOC*, 2009.

[5] K. Chaudhuri and C. Monteleoni. Privacy-preserving logistic regression. In *NIPS*, pages 289–296, 2008.

[6] K. Chaudhuri, C. Monteleoni, and A.D. Sarwate. Differentially private empirical risk minimization. 2010.

[7] C. Dwork, F. McSherry, K. Nissim, and A. Smith. Calibrating noise to sensitivity in private data analysis. In *TCC*, pages 265–284, 2006.

[8] F. McSherry and K. Talwar. Differential privacy via mechanism design. In *FOCS*, 2007.

[9] M.I. Jordan, Z. Ghahramani, T. Jaakkola, and L.K. Saul. An introduction to variational methods for graphical models. *Machine Learning*, 37(2):183–233, 1999.

[10] F. McSherry. Privacy integrated queries. In *ACM SIGMOD*, 2009.

[11] A. Asuncion and D.J. Newman. UCI machine learning repository, 2007.

